# Random function priors for exchangeable arrays with applications to graphs and relational data

**James Robert Lloyd**
Department of Engineering
University of Cambridge

**Peter Orbanz**
Department of Statistics
Columbia University

**Zoubin Ghahramani**
Department of Engineering
University of Cambridge

**Daniel M. Roy**
Department of Engineering
University of Cambridge

## Abstract

A fundamental problem in the analysis of structured relational data like graphs, networks, databases, and matrices is to extract a summary of the common structure underlying relations between individual entities. Relational data are typically encoded in the form of arrays; invariance to the ordering of rows and columns corresponds to exchangeable arrays. Results in probability theory due to Aldous, Hoover and Kallenberg show that exchangeable arrays can be represented in terms of a random measurable function which constitutes the natural model parameter in a Bayesian model. We obtain a flexible yet simple Bayesian nonparametric model by placing a Gaussian process prior on the parameter function. Efficient inference utilises elliptical slice sampling combined with a random sparse approximation to the Gaussian process. We demonstrate applications of the model to network data and clarify its relation to models in the literature, several of which emerge as special cases.

## 1 Introduction

Structured relational data arises in a variety of contexts, including graph-valued data [e.g. 1, 5], micro-array data, tensor data [e.g. 27] and collaborative filtering [e.g. 21]. This data is typified by expressing relations between 2 or more objects (e.g. friendship between a pair of users in a social network). Pairwise relations can be represented by a 2-dimensional array (a matrix); more generally, relations between $d$-tuples are recorded as $d$-dimensional arrays ($d$-arrays). We consider Bayesian models of infinite 2-arrays $(X_{ij})_{i,j\in\mathbb{N}}$, where entries $X_{ij}$ take values in a space $\mathcal{X}$. Each entry $X_{ij}$ describes the relation between objects $i$ and $j$. Finite samples—relational measurements for $n$ objects—are $n \times n$-arrays. As the sample size increases, the data aggregates into a larger and larger array. Graph-valued data, for example, corresponds to the case $\mathcal{X} = \{0, 1\}$. In collaborative filtering problems, the set of objects is subdivided into two disjoint sets, e.g., users and items.

Latent variable models for such data explain observations by means of an underlying structure or summary, such as a low-rank approximation to an observed array or an embedding into a Euclidean space. This structure is formalized as a latent (unobserved) variable. Examples include matrix factorization [e.g. 4, 21], non-linear generalisations [e.g. 12, 27, 28], block modelling [e.g. 1, 10], latent distance modelling [e.g. 5] and many others [e.g. 14, 17, 20].

Hoff [4] first noted that a number of parametric latent variable models for relational data are exchangeable—an applicable assumption whenever the objects in the data have no natural ordering e.g., users in a social network or products in ratings data—and can be cast into the common functional form guaranteed to exist by results in probability theory. Building on this connection,

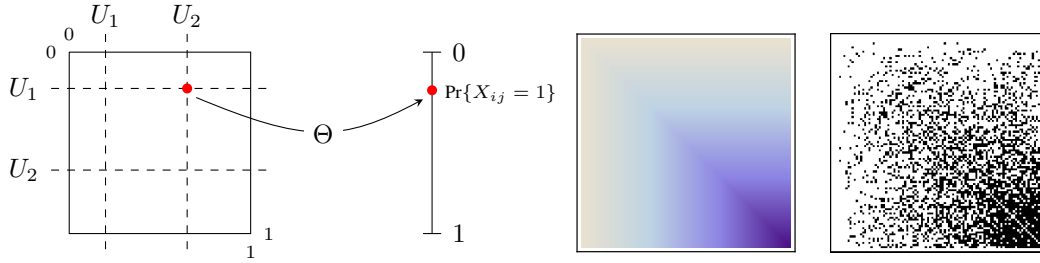

Figure 1: *Left:* The distribution of any exchangeable random graph with vertex set $\mathbb{N}$ and edges $E = (X_{ij})_{i,j\in\mathbb{N}}$ can be characterised by a random function $\Theta : [0,1]^2 \to [0,1]$. Given $\Theta$, a graph can be sampled by generating a uniform random variable $U_i$ for each vertex $i$, and sampling edges as $X_{ij} \sim \text{Bernoulli}(\Theta(U_i, U_j))$. *Middle:* A heat map of an example function $\Theta$. *Right:* A $100 \times 100$ symmetric adjacency matrix sampled from $\Theta$. Only unordered index pairs $X_{ij}$ are sampled in the symmetric case. Rows and columns have been ordered by increasing value of $U_i$, rather than $i$.

we consider nonparametric models for graphs and arrays. Results of Aldous [2], Hoover [6] and Kallenberg [7] show that random arrays that satisfy an exchangeability property can be represented in terms of a random function. These representations have been further developed in discrete analysis for the special case of graphs [13]; this case is illustrated in Fig. 1. The results can be regarded as a generalization of de Finetti's theorem to array-valued data. Their implication for Bayesian modeling is that *we can specify a prior for an exchangeable random array model by specifying a prior on (measurable) functions.* The prior is a distribution on the space of all functions that can arise in the representation result, and the dimension of this space is infinite. A prior must therefore be *nonparametric* to have reasonably large support since a parametric prior concentrates on a finite-dimensional subset. In the following, we model the representing function explicitly using a nonparametric prior.

## 2  Background: Exchangeable graphs and arrays

A fundamental component of every Bayesian model is a random variable $\Theta$, the parameter of the model, which decouples the data. De Finetti's theorem [9] characterizes this parameter for random sequences: Let $X_1, X_2, \ldots$ be an infinite sequence of random variables, each taking values in a common space $\mathcal{X}$. A sequence is called *exchangeable* if its joint distribution is invariant under arbitrary permutation of the indices, i.e., if

$$(X_1, X_2, \ldots) \overset{\mathrm{d}}{=} (X_{\pi(1)}, X_{\pi(2)}, \ldots) \qquad \text{for all } \pi \in \mathbb{S}_\infty \ . \tag{2.1}$$

Here, $\overset{\mathrm{d}}{=}$ denotes equality in distribution, and $\mathbb{S}_\infty$ is the set of all permutations of $\mathbb{N}$ that permute a finite number of elements. De Finetti's theorem states that, $(X_i)_{i\in\mathbb{N}}$ is exchangeable if and only if there exists a random probability measure $\Theta$ on $\mathcal{X}$ such that $X_1, X_2, \ldots \,|\, \Theta \sim_{\text{iid}} \Theta$, i.e., conditioned on $\Theta$, the observations are independent and $\Theta$-distributed. From a statistical perspective, $\Theta$ represents common structure in the observed data—and thus a natural target of statistical inference—whereas $P[X_i|\Theta]$ captures remaining, independent randomness in each observation.

### 2.1  De Finetti-type representations for random arrays

To specify Bayesian models for graph- or array-valued data, we need a suitable counterpart to de Finetti's theorem that is applicable when the random sequences in (2.1) are substituted by random arrays $X = (X_{ij})_{i,j\in\mathbb{N}}$. For such data, the invariance assumption (2.1) applied to all elements of $X$ is typically too restrictive: In the graph case $X_{ij} \in \{0,1\}$, for example, the probability of $X$ would then depend only on the proportion of edges present in the graph, but not on the graph structure. Instead, we define exchangeability of random 2-arrays in terms of the *simultaneous* application of a permutation to rows and columns. More precisely:

**Definition 2.1.** An array $X = (X_{ij})_{i,j\in\mathbb{N}}$ is called an *exchangeable array* if

$$(X_{ij}) \overset{\mathrm{d}}{=} (X_{\pi(i)\pi(j)}) \qquad \text{for every } \pi \in \mathbb{S}_\infty \ . \tag{2.2}$$

Since this weakens the hypothesis (2.1) by demanding invariance only under a subset of all permutations of $\mathbb{N}^2$—those of the form $(i, j) \mapsto (\pi(i), \pi(j))$—we can no longer expect de Finetti's theorem to hold. The relevant generalization of the de Finetti theorem to this case is the following:

**Theorem 2.2** (Aldous, Hoover). *A random 2-array $(X_{ij})$ is exchangeable if and only if there is a random (measurable) function $F : [0, 1]^3 \to \mathcal{X}$ such that*

$$(X_{ij}) \stackrel{d}{=} (F(U_i, U_j, U_{ij})). \tag{2.3}$$

*for every collection $(U_i)_{i \in \mathbb{N}}$ and $(U_{ij})_{i \leq j \in \mathbb{N}}$ of i.i.d. Uniform$[0, 1]$ random variables, where $U_{ji} = U_{ij}$ for $j < i \in \mathbb{N}$.*

## 2.2 Random graphs

The graph-valued data case $\mathcal{X} = \{0, 1\}$ is of particular interest. Here, the array $X$, interpreted as an adjacency matrix, specifies a random graph with vertex set $\mathbb{N}$. For undirected graphs, $X$ is symmetric. We call a random graph exchangeable if $X$ satisfies (2.2).

For undirected graphs, the representation (2.3) simplifies further: there is a random function $\Theta : [0, 1]^2 \to [0, 1]$, symmetric in its arguments, such that

$$F(U_i, U_j, U_{ij}) := \begin{cases} 1 & \text{if } U_{ij} < \Theta(U_i, U_j) \\ 0 & \text{otherwise} \end{cases} \tag{2.4}$$

satisfies (2.3). Each variable $U_i$ is associated with a vertex, each variable $U_{ij}$ with an edge. The representation (2.4) is equivalent to the sampling scheme

$$U_1, U_2, \ldots \sim_{\text{iid}} \text{Uniform}[0, 1] \qquad \text{and} \qquad X_{ij} = X_{ji} \sim \text{Bernoulli}(\Theta(U_i, U_j)) , \tag{2.5}$$

which is illustrated in Fig. 1.

Recent work in discrete analysis shows that any symmetric measurable function $[0, 1]^2 \to [0, 1]$ can be regarded as a (suitably defined) limit of adjacency matrices of graphs of increasing size [13]—intuitively speaking, as the number of rows and columns increases, the array in Fig. 1 (right) converges to the heat map in Fig. 1 (middle) (up to a reordering of rows and columns).

## 2.3 The general case: $d$-arrays

Theorem 2.2 can in fact be stated in a more general setting than 2-arrays, namely for random $d$-arrays, which are collections of random variables of the form $(X_{i_1 \ldots i_d})_{i_1, \ldots, i_d \in \mathbb{N}}$. Thus, a sequence is a 1-array, a matrix a 2-array. A $d$-array can be interpreted as an encoding of a relation between $d$-tuples. In this general case, an analogous theorem holds, but the random function $F$ in (2.3) is in general more complex: In addition to the collections $U_{\{i\}}$ and $U_{\{ij\}}$ of uniform variables, the representation requires an additional collection $U_{\{i_j\}_{j \in I}}$ for every non-empty subset $I \subseteq \{1, \ldots, d\}$; e.g., $U_{\{i_1 i_3 i_4\}}$ for $d \geq 4$ and $I = \{1, 3, 4\}$. The representation (2.3) is then substituted by

$$F : [0, 1]^{2^d - 1} \longrightarrow \mathcal{X} \qquad \text{and} \qquad (X_{i_1, \ldots, i_d}) \stackrel{d}{=} (F(U_{I_1}, \ldots, U_{I_{(2^d - 1)}})) . \tag{2.6}$$

For $d = 1$, we recover a version of de Finetti's theorem. For a discussion of convergence properties of general arrays similar to those sketched above for random graphs, see [3].

Because we do not explicitly consider the case $d > 2$ in our experiments, we restrict our presentation of the model to the 2-array-valued case for simplicity. We note, however, that the model and inference algorithms described in the following extend immediately to general $d$-array-valued data.

## 3 Model

To define a Bayesian model for exchangeable graphs or arrays, we start with Theorem 2.2: A distribution on exchangeable arrays can be specified by a distribution on measurable functions $[0, 1]^3 \to \mathcal{X}$. We decompose the function $F$ into two functions $\Theta : [0, 1]^2 \to \mathcal{W}$ and $H : [0, 1] \times \mathcal{W} \to \mathcal{X}$ for a suitable space $\mathcal{W}$, such that

$$(X_{ij}) \stackrel{d}{=} (F(U_i, U_j, U_{ij})) = (H(U_{ij}, \Theta(U_i, U_j))) . \tag{3.1}$$

Such a decomposition always exists—trivially, choose $\mathcal{W} = [0, 1]^2$. The decomposition introduces a natural hierarchical structure. We initially sample a random function $\Theta$—the model parameter in terms of Bayesian statistics—which captures the structure of the underlying graph or array. The $(U_i)$ then represent attributes of nodes or objects and $H$ and the array $(U_{ij})$ model the remaining noise in the observed relations.

**Model definition.** For the purpose of defining a Bayesian model, we will model $\Theta$ as a continuous function with a Gaussian process prior. More precisely, we take $\mathcal{W} = \mathbb{R}$ and consider a zero-mean Gaussian process prior on $\mathbf{C}_\mathcal{W} := \mathbf{C}([0, 1]^2, \mathcal{W})$, the space of continuous functions from $[0, 1]^2$ to $\mathcal{W}$, with kernel function $\kappa : [0, 1]^2 \times [0, 1]^2 \to \mathcal{W}$. The full generative model is then:

$$
\begin{aligned}
\Theta &\sim \mathcal{GP}(0, \kappa) \\
U_1, U_2, \dots &\sim_{\text{iid}} \text{Uniform}[0, 1] \\
X_{ij} \,|\, W_{ij} &\sim P[\,.\,|W_{ij}] \qquad \text{where } W_{ij} = \Theta(U_i, U_j) \,.
\end{aligned} \tag{3.2}
$$

The parameter space of our the model is the infinite-dimensional space $\mathbf{C}_\mathcal{W}$. Hence, the model is nonparametric.

Graphs and real-valued arrays require different choices of $P$. In either case, the model first generates the latent array $W = (W_{ij})$. Observations are then generated as follows:

| Observed data | Sample space | $P[X_{ij} \in \,.\,|W_{ij}]$ |
|---|---|---|
| Graph | $\mathcal{X} = \{0, 1\}$ | Bernoulli$(\phi(W_{ij}))$ |
| Real array | $\mathcal{X} = \mathbb{R}$ | Normal$(W_{ij}, \sigma_\mathcal{X}^2)$ |

where $\phi$ is the logistic function, and $\sigma_\mathcal{X}^2$ is a noise variance parameter.

The Gaussian process prior favors smooth functions, which will in general result in more interpretable latent space embeddings. Inference in Gaussian processes is a well-understood problem, and the choice of a Gaussian prior allows us to leverage the full range of inference methods available for these models.

**Discussion of modeling assumptions.** In addition to exchangeability, our model assumes (i) that the function $\Theta$ is continuous—which implies measurability as in Theorem 2.2 but is a stronger requirement—and (ii) that its law is Gaussian. Exchangeable, undirected graphs are always representable using a Bernoulli distribution for $P[X_{ij} \in \,.\,|W_{ij}]$. Hence, in this case, (i) and (ii) are indeed the only assumptions imposed by the model. In the case of real-valued matrices, the model additionally assumes that the function $H$ in (3.1) is of the form

$$
H(U_{ij}, \Theta(U_i, U_j)) \stackrel{\text{d}}{=} \Theta(U_i, U_j) + \varepsilon_{ij} \qquad \text{where} \qquad \varepsilon_{ij} \sim_{\text{iid}} \text{Normal}(0, \sigma) \,. \tag{3.3}
$$

Another rather subtle assumption arises implicitly when the array $X$ is not symmetric, i.e., not guaranteed to satisfy $X_{ij} = X_{ji}$, for example, if $X$ is a directed graph: In Theorem 2.2, the array $(U_{ij})$ is symmetric even if $X$ is not. The randomness in $U_{ij}$ accounts for both $X_{ij}$ and $X_{ji}$ which means the conditional variables $X_{ij}|W_{ij}$ and $X_{ji}|W_{ji}$ are dependent, and a precise representation would have to sample $(X_{ij}, X_{ji})|W_{ij}, W_{ji}$ jointly, a fact our model neglects in (3.2). However, it can be shown that any exchangeable array can be arbitrarily well approximated by arrays which treat $X_{ij}|W_{ij}$ and $X_{ji}|W_{ji}$ as independent [8, Thm. 2].

**Remark 3.1** (Dense vs. sparse data)**.** The methods described here address random arrays that are *dense*, i.e., as the size of an $n \times n$ array increases the number of non-zero entries grows as $O(n^2)$. Network data is typically *sparse*, with $O(n)$ non-zero entries. Density is an immediate consequence of Theorem 2.2: For graph data the asymptotic proportion of present edges is $p := \int \Theta(x, y) dx dy$, and the graph is hence either empty (for $p = 0$) or dense (since $O(pn^2) = O(n^2)$). Analogous representation theorems for sparse random graphs are to date an open problem in probability.

## 4   Related work

Our model has some noteworthy relations to the Gaussian process latent variable model (GPLVM); a dimensionality-reduction technique [e.g. 11]. GPLVMs can be applied to 2-arrays, but doing so makes the assumption that either the rows or the columns of the random array are independent [12]. In terms of our model, this corresponds to choosing kernels of the form $\kappa_U \otimes \delta$, where $\otimes$ represents

a tensor product[1]and $\delta$ represents an 'identity' kernel (i.e., the corresponding kernel matrix is the identity matrix). From this perspective, the application of our model to exchangeable real-valued arrays can be interpreted as a form of co-dimensionality reduction.

For graph data, a related parametric model is the eigenmodel of Hoff [4]. This model, also justified by exchangeability arguments, approximates an array with a bilinear form, followed by some link function and conditional probability distribution.

Available nonparametric models include the infinite relational model (IRM) [10], latent feature relational model (LFRM) [14], infinite latent attribute model (ILA) [17] and many others. A recent development is the sparse matrix-variate Gaussian process blockmodel (SMGB) of Yan *et al.* [28]. Although not motivated in terms of exchangeability, this model does not impose an independence assumptions on either rows or columns, in contrast to the GPLVM. The model uses kernels of the form $\kappa_1 \otimes \kappa_2$; our work suggests that it may not be necessary to impose tensor product structure, which allows for inference with improved scaling. Roy and Teh [20] present a nonparametric Bayesian model of relational data that approximates $\Theta$ by a piece-wise constant function with a specific hierarchical structure, which is called a Mondrian process in [20].

Some examples of the various available models can be succinctly summarized as follows:

| Graph data | | | |
|---|---|---|---|
| Random function model | $\Theta$ | $\sim$ | $\mathcal{GP}\left(0,\kappa\right)$ |
| Latent class [26] | $W_{ij}$ | $=$ | $m_{U_i U_j}$ where $U_i \in \{1,\ldots,K\}$ |
| IRM [10] | $W_{ij}$ | $=$ | $m_{U_i U_j}$ where $U_i \in \{1,\ldots,\infty\}$ |
| Latent distance [5] | $W_{ij}$ | $=$ | $-|U_i - U_j|$ |
| Eigenmodel [4] | $W_{ij}$ | $=$ | $U_i'\Lambda U_j$ |
| LFRM [14] | $W_{ij}$ | $=$ | $U_i'\Lambda U_j$ where $U_i \in \{0,1\}^\infty$ |
| ILA [17] | $W_{ij}$ | $=$ | $\sum_d \mathbb{I}_{U_{id}}\mathbb{I}_{U_{jd}}\Lambda^{(d)}_{U_{id}U_{jd}}$ where $U_i \in \{0,\ldots,\infty\}^\infty$ |
| SMGB [28] | $\Theta$ | $\sim$ | $\mathcal{GP}\left(0,\kappa_1 \otimes \kappa_2\right)$ |
| Real-valued array data | | | |
| Random function model | $\Theta$ | $\sim$ | $\mathcal{GP}\left(0,\kappa\right)$ |
| Mondrian process based [20] | $\Theta$ | $=$ | piece-wise constant random function |
| PMF [21] | $W_{ij}$ | $=$ | $U_i'V_j$ |
| GPLVM [12] | $\Theta$ | $\sim$ | $\mathcal{GP}\left(0,\kappa \otimes \delta\right)$ |

# 5 Posterior computation

We describe Markov Chain Monte Carlo (MCMC) algorithms for generating approximate samples from the posterior distribution of the model parameters given a partially observed array. Most importantly, we describe a random subset-of-regressors approximation that scales to graphs with hundreds of nodes. Given the relatively straightforward nature of the proposed algorithms and approximations, we refer the reader to other papers whenever appropriate.

## 5.1 Latent space and kernel

Theorem 2.2 is not restricted to the use of uniform distributions for the variables $U_i$ and $U_{ij}$. The proof remains unchanged if one replaces the uniform distributions with any non-atomic probability measure on a Borel space. For the purposes of inference, normal distributions are more convenient, and we henceforth use $U_1, U_2, \ldots \sim_{\text{iid}} \mathcal{N}(0, I_r)$ for some integer $r$.

Since we focus on undirected graphical data, we require the symmetry condition $W_{ij} = W_{ji}$. This can be achieved by constructing the kernel function in the following way

$$\kappa(\xi_1, \xi_2) = \frac{1}{2}\big(\bar{\kappa}(\xi_1, \xi_2) + \bar{\kappa}(\xi_1, \bar{\xi}_2)\big) + \sigma^2 I \qquad \text{(Symmetry + noise)} \qquad (5.1)$$

$$\bar{\kappa}(\xi_1, \xi_2) = s^2 \exp(-|\xi_1 - \xi_2|^2/(2\ell^2)) \qquad \text{(RBF kernel)} \qquad (5.2)$$

where $\xi_k = (U_{i_k}, U_{j_k})$, $\bar{\xi}_k = (U_{j_k}, U_{i_k})$ and $s, \ell, \sigma$ represent a scale factor, length scale and noise respectively (see [e.g. 19] for a discussion of kernel functions). We collectively denote the kernel parameters by $\psi$.

## 5.2 Sampling without approximating the model

In the simpler case of a real-valued array $X$, we construct an MCMC algorithm over the variables $(U, \psi, \sigma_X)$ by repeatedly slice sampling [16] from the conditional distributions

$$\psi_i \,|\, \psi_{-i}, \sigma_X, U, X \qquad \sigma_X \,|\, \psi, U, X \qquad \text{and} \qquad U_j \,|\, U_{-j}, \psi, \sigma_X, X \qquad (5.3)$$

where $\sigma_X$ is the noise variance parameter used when modelling real valued data introduced in section 3. Let $\mathrm{N} = |U_{\{i\}}|$ denote the number of rows in the observed array, let $\xi$ be the set of all pairs $(U_i, U_j)$ for all observed relations $X_{ij}$, let $\mathrm{O} = |\xi|$ denote the number of observed relations, and let $K$ represent the $\mathrm{O} \times \mathrm{O}$ kernel matrix between all points in $\xi$. Changes to $\psi$ affect every entry in the kernel matrix $K$ and so, naively, the computation of the Gaussian likelihood of $X$ takes $\mathcal{O}(\mathrm{O}^3)$ time. The cubic dependence on O seems unavoidable, and thus this naive algorithm is unusable for all but small data sets.

## 5.3 A random subset-of-regressor approximation

To scale the method to larger graphs, we apply a variation of a method known as Subsets-of-Regressors (SoR) [22, 23, 25]. (See [18] for an excellent survey of this and other sparse approximations.) The SoR approximation replaces the infinite dimensional GP with a finite dimensional approximation. Our approach is to treat both the inputs and outputs of the GP as latent variables.

In particular, we introduce $k$ Gaussian distributed pseudoinputs $\eta = (\eta_1, \ldots, \eta_k)$ and define target values $T_j = \Theta(\eta_j)$. Writing $K_{\eta\eta}$ for the kernel matrix formed from the pseudoinputs $\eta$, we have

$$(\eta_i) \sim_{\text{iid}} \mathcal{N}(0, I_{2r}) \qquad \text{and} \qquad T \,|\, \eta \sim \mathcal{N}(0, K_{\eta\eta}). \qquad (5.4)$$

The idea of the SoR approximation is to replace $W_{ij}$ with the posterior mean conditioned on $(\eta, T)$,

$$W = K_{\xi\eta} K_{\eta\eta}^{-1} T, \qquad (5.5)$$

where $K_{\xi\eta}$ is the kernel matrix between the latent embeddings $\xi$ and the pseudoinputs $\eta$. By considering random pseudoinputs, we construct an MCMC analogue of the techniques proposed in [24].

The conditional distribution $T \,|\, U, \eta, \psi, (\sigma_X), X$ is amenable to elliptical slice sampling [15]. All other random parameters, including the $(U_i)$, can again be sampled from their full conditional distributions using slice sampling. The sampling algorithms require that one computes expressions involving (5.5). As a result they cost at most $\mathcal{O}(k^3\mathrm{O})$ time.

# 6 Experiments

We evaluate the model on three different network data sets. Two of these data sets—the high school and NIPS co-authorship data—have been extensively analyzed in the literature. The third data set, a protein interactome, was previously noted by Hoff [4] to be of interest since it exhibits both block structure and transitivity.

| Data set | Recorded data | Vertices | Reference |
|---|---|---|---|
| High school | high school social network | 90 | e.g. [4] |
| NIPS | densely connected subset of coauthorship network | 234 | e.g. [14] |
| Protein | protein interactome | 230 | e.g. [4] |

We compare performance of our model on these data sets to three other models, probabilistic matrix factorization (PMF) [21], Hoff's eigenmodel, and the GPLVM (see also Sec. 4). The models are chosen for comparability, since they all embed nodes into a Euclidean latent space. Experiments for all three models were performed using reference implementations by the respective authors.[2]

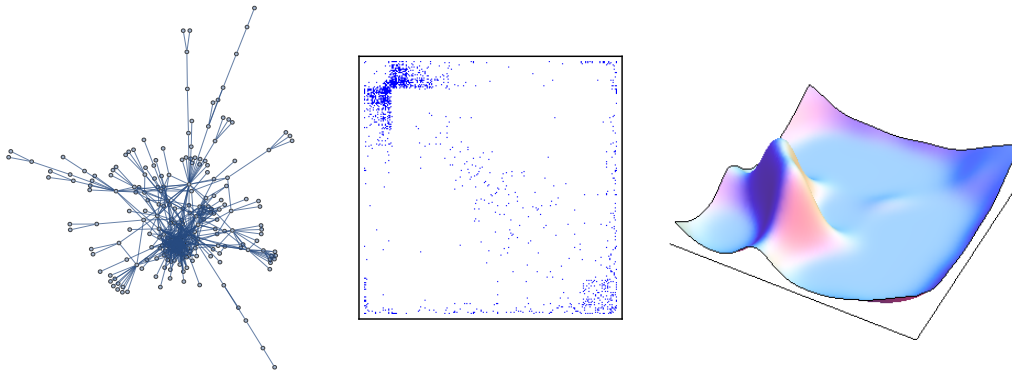

Figure 2: Protein interactome data. *Left:* Interactome network. *Middle:* Sorted adjacency matrix. The network exhibits stochastic equivalence (visible as block structure in the matrix) and homophily (concentration of points around the diagonal). *Right:* Maximum a posteriori estimate of the function $\Theta$, corresponding to the function in Fig. 1 (middle).

| Model | Method | Iterations [burn-in] | Algorithm parameters |
|---|---|---|---|
| PMF [21] | stochastic gradient | 1000 | author defaults |
| Eigenmodel [4] | MCMC | 10000 [250] | author defaults |
| GPLVM [12] | stochastic gradient | 20 sweeps | author defaults |
| Random function model | MCMC | 1000 [200] | (see below) |

We use standard normal priors on the latent variables $U$ and pseudo points $\eta$, and log normal priors for kernel parameters. Parameters are chosen to favor slice sampling acceptance after a reasonable number of iterations, as evaluated over a range of data sets, summarized in the table on the right. Balancing computational demands, we sampled $T$ 50 times per iteration whilst all other variables were sampled once per iteration.

| | log mean | std | width |
|---|---|---|---|
| length scale | 1 | 0.5 | 0.5 |
| scale factor | 2 | 0.5 | 0.5 |
| target noise | 0.1 | 0.5 | 0.1 |
| $U$ | - | - | 4 |
| $\eta$ | - | - | 2 |

We performed 5-fold cross validation, predicting links in a held out partition given 4 others. Where the models did not restrict their outputs to values between 0 and 1, we truncated any predictions lying outside this range. The following table reports average AUC (area under receiver operating characteristic) for the various models, with numbers for the top performing model set in bold. Significance of results is evaluated by means of a $t$-test with a $p$-value of 0.05; results for models not distinguishable from the top performing model in terms of this $t$-test are also set in bold.

| | AUC results | | | | | | | | |
|---|---|---|---|---|---|---|---|---|---|
| Data set | High school | | | NIPS | | | Protein | | |
| Latent dimensions | 1 | 2 | 3 | 1 | 2 | 3 | 1 | 2 | 3 |
| PMF | 0.747 | 0.792 | 0.792 | 0.729 | 0.789 | 0.820 | 0.787 | 0.810 | 0.841 |
| Eigenmodel | 0.742 | 0.806 | 0.806 | 0.789 | 0.818 | 0.845 | 0.805 | 0.866 | 0.882 |
| GPLVM | 0.744 | 0.775 | 0.782 | 0.888 | 0.876 | 0.883 | 0.877 | 0.883 | 0.873 |
| RFM | **0.815** | **0.827** | **0.820** | **0.907** | **0.914** | **0.919** | **0.903** | **0.910** | **0.912** |

The random function model outperforms the other models in *all* tests. We also note that in all experiments, a single latent dimension suffices to achieve better performance, even when the other models use additional latent dimensions.

The posterior distribution of $\Theta$ favors functions defining random array distributions that explain the data well. In this sense, our model fits a probability distribution. The standard inference methods for GPLVM and PMF applied to relational data, in contrast, are designed to fit mean squared error, and should therefore be expected to show stronger performance under a mean squared error metric. As the following table shows, this is indeed the case.

RMSE results

| Data set | High school | | | NIPS | | | Protein | | |
|---|---|---|---|---|---|---|---|---|---|
| Latent dimensions | 1 | 2 | 3 | 1 | 2 | 3 | 1 | 2 | 3 |
| PMF | 0.245 | 0.242 | 0.240 | 0.141 | 0.135 | 0.130 | 0.151 | 0.142 | 0.139 |
| Eigenmodel | 0.244 | 0.238 | **0.236** | 0.141 | 0.132 | 0.124 | 0.149 | 0.142 | **0.138** |
| GPLVM | 0.244 | 0.241 | 0.239 | **0.112** | **0.109** | **0.106** | **0.139** | 0.137 | 0.138 |
| RFM | **0.239** | **0.234** | 0.235 | 0.114 | 0.111 | 0.110 | 0.138 | **0.136** | **0.136** |

An arguably more suitable metric is comparison in terms of conditional edge probability i.e., $P(X_{\{ij\}} \,|\, W_{\{ij\}})$ for all $i, j$ in the held out data. These cannot, however, be computed in a meaningful manner for models such as PMF and GPLVM, which assign a Gaussian likelihood to data. The next table hence reports only comparisons to the eigenmodel.

Negative log conditional edge probability[3]

| Data set | High school | | | NIPS | | | Protein | | |
|---|---|---|---|---|---|---|---|---|---|
| Latent dimensions | 1 | 2 | 3 | 1 | 2 | 3 | 1 | 2 | 3 |
| Eigenmodel | 220 | 210 | **200** | 88 | 81 | 75 | 96 | 92 | 86 |
| RFM | **205** | **199** | 201 | **65** | **57** | **56** | **78** | **75** | **75** |

**Remark 6.1** (Model complexity and lengthscales). Figure 2 provides a visualisation of $\Theta$ when modeling the protein interactome data using 1 latent dimension. The likelihood of the smooth peak is sensitive to the lengthscale of the Gaussian process representation of $\Theta$. A Gaussian process prior introduces the assumption that $\Theta$ is continuous. Continuous functions are dense in the space of measurable functions, i.e., any measurable function can be arbitrarily well approximated by a continuous one. The assumption of continuity is therefore not restrictive, but rather the lengthscale of the Gaussian process determines the complexity of the model a priori. The nonparametric prior placed on $\Theta$ allows the posterior to approximate any function if supported by the data, but by sampling the lengthscale we allow the model to quickly select an appropriate level of complexity.

# 7 Discussion and conclusions

There has been a tremendous amount of research into modelling matrices, arrays, graphs and relational data, but nonparametric Bayesian modeling of such data is essentially uncharted territory. In most modelling circumstances, the assumption of exchangeability amongst data objects is natural and fundamental to the model. In this case, the representation results [2, 6, 7] precisely map out the scope of possible Bayesian models for exchangeable arrays: Any such model can be interpreted as a prior on random measurable functions on a suitable space.

Nonparametric Bayesian statistics provides a number of possible priors on random functions, but the Gaussian process and its modifications are the only well-studied model for almost surely continuous functions. For this choice of prior, our work provides a general and simple modeling approach that can be motivated directly by the relevant representation results. The model results in both interpretable representations for networks, such as a visualisation of a protein interactome, and has competitive predictive performance on benchmark data.

### Acknowledgments

The authors would like to thank David Duvenaud, David Knowles and Konstantina Palla for helpful discussions. PO was supported by an EPSRC Mathematical Sciences Postdoctoral Research Fellowship (EP/I026827/1). ZG is supported by EPSRC grant EP/I036575/1. DMR is supported by a Newton International Fellowship and Emmanuel College.

## Footnotes

[1]We define the tensor product of kernel functions as follows: $(\kappa_U \otimes \kappa_V)((u_1,v_1),(u_2,v_2)) = \kappa_U(u_1,u_2) \times \kappa_V(v_1,v_2)$.

[2]Implementations are available for PMF at http://www.mit.edu/~rsalakhu/software.html; for the eignmodel at http://cran.r-project.org/src/contrib/Descriptions/eigenmodel.html; and for the GPLVM at http://www.cs.man.ac.uk/~neill/collab/ .

[3]The precise calculation implemented is $-\log(P(X_{\{ij\}} \,|\, W_{\{ij\}})) \times 1000$ / (Number of held out edges).

# References

[1] Airoldi, E. M., Blei, D. M., Fienberg, S. E., and Xing, E. P. (2008). Mixed Membership Stochastic Block-models. *Journal of Machine Learning Research (JMLR)*, **9**, 1981–2014.

[2] Aldous, D. J. (1981). Representations for partially exchangeable arrays of random variables. *Journal of Multivariate Analysis*, **11**(4), 581–598.

[3] Aldous, D. J. (2010). More uses of exchangeability: Representations of complex random structures. In *Probability and Mathematical Genetics: Papers in Honour of Sir John Kingman*.

[4] Hoff, P. D. (2007). Modeling homophily and stochastic equivalence in symmetric relational data. In *Advances in Neural Information Processing Systems (NIPS)*, volume 20, pages 657–664.

[5] Hoff, P. D., Raftery, A. E., and Handcock, M. S. (2002). Latent Space Approaches to Social Network Analysis. *Journal of the American Statistical Association*, **97**(460), 1090–1098.

[6] Hoover, D. N. (1979). Relations on probability spaces and arrays of random variables. Technical report, Institute for Advanced Study, Princeton.

[7] Kallenberg, O. (1992). Symmetries on random arrays and set-indexed processes. *Journal of Theoretical Probability*, **5**(4), 727–765.

[8] Kallenberg, O. (1999). Multivariate Sampling and the Estimation Problem for Exchangeable Arrays. *Journal of Theoretical Probability*, **12**(3), 859–883.

[9] Kallenberg, O. (2005). *Probabilistic Symmetries and Invariance Principles*. Springer.

[10] Kemp, C., Tenenbaum, J., Griffiths, T., Yamada, T., and Ueda, N. (2006). Learning systems of concepts with an infinite relational model. In *Proceedings of the National Conference on Artificial Intelligence*, volume 21.

[11] Lawrence, N. D. (2005). Probabilistic non-linear principal component analysis with Gaussian process latent variable models. *Journal of Machine Learning Research (JMLR)*, **6**, 1783–1816.

[12] Lawrence, N. D. and Urtasun, R. (2009). Non-linear matrix factorization with Gaussian processes. In *Proceedings of the International Conference on Machine Learning (ICML)*, pages 1–8. ACM Press.

[13] Lovász, L. and Szegedy, B. (2006). Limits of dense graph sequences. *Journal of Combinatorial Theory Series B*, **96**, 933–957.

[14] Miller, K. T., Griffiths, T. L., and Jordan, M. I. (2009). Nonparametric latent feature models for link prediction. *Advances in Neural Information Processing Systems (NIPS)*, pages 1276–1284.

[15] Murray, I., Adams, R. P., and Mackay, D. J. C. (2010). Elliptical slice sampling. *Journal of Machine Learning Research (JMLR)*, **9**, 541–548.

[16] Neal, R. M. (2003). Slice sampling. *The Annals of Statistics*, **31**(3), 705–767. With discussions and a rejoinder by the author.

[17] Palla, K., Knowles, D. A., and Ghahramani, Z. (2012). An Infinite Latent Attribute Model for Network Data. In *Proceedings of the International Conference on Machine Learning (ICML)*.

[18] Quiñonero Candela, J. and Rasmussen, C. E. (2005). A unifying view of sparse approximate gaussian process regression. *Journal of Machine Learning Research (JMLR)*, **6**, 1939–1959.

[19] Rasmussen, C. E. and Williams, C. K. I. (2006). *Gaussian Processes for Machine Learning*. MIT Press.

[20] Roy, D. M. and Teh, Y. W. (2009). The Mondrian process. In *Advances in Neural Information Processing Systems (NIPS)*.

[21] Salakhutdinov, R. (2008). Probabilistic Matrix Factorisation. In *Advances in neural information processing systems (NIPS)*.

[22] Silverman, B. W. (1985). Some aspects of the spline smoothing approach to non-parametric regression curve fitting. *Journal of the Royal Statistical Society. Series B (Methodological)*, **47**(1), 1–52.

[23] Smola, A. J. and Bartlett, P. (2001). Sparse greedy gaussian process regression. In *Advances in Neural Information Processing Systems (NIPS)*. MIT Press.

[24] Titsias, M. K. and Lawrence, N. D. (2008). Efficient sampling for Gaussian process inference using control variables. In *Advances in Neural Information Processing Systems (NIPS)*, pages 1681–1688.

[25] Wahba, G., Lin, X., Gao, F., Xiang, D., Klein, R., and Klein, B. (1999). The bias-variance tradeoff and the randomized gacv. In *Advances in Neural Information Processing Systems (NIPS)*.

[26] Wang, Y. J. and Wong, G. Y. (1987). Stochastic Blockmodels for Directed Graphs. *Journal of the American Statistical Association*, **82**(397), 8–19.

[27] Xu, Z., Yan, F., and Qi, Y. (2012). Infinite Tucker Decomposition: Nonparametric Bayesian Models for Multiway Data Analysis. In *Proceedings of the International Conference on Machine Learning (ICML)*.

[28] Yan, F., Xu, Z., and Qi, Y. (2011). Sparse matrix-variate Gaussian process blockmodels for network modeling. In *Proceedings of the International Conference on Uncertainty in Artificial Intelligence (UAI)*.

